# Information Rates and Optimal Decoding in Large Neural Populations

**Kamiar Rahnama Rad    Liam Paninski**
Department of Statistics, Columbia University
{kamiar,liam}@stat.columbia.edu
`http://www.stat.columbia.edu/˜liam/research/pubs/kamiar-ss-info.pdf`

## Abstract

Many fundamental questions in theoretical neuroscience involve optimal decoding and the computation of Shannon information rates in populations of spiking neurons. In this paper, we apply methods from the asymptotic theory of statistical inference to obtain a clearer analytical understanding of these quantities. We find that for large neural populations carrying a finite total amount of information, the full spiking population response is asymptotically as informative as a single observation from a Gaussian process whose mean and covariance can be characterized explicitly in terms of network and single neuron properties. The Gaussian form of this asymptotic sufficient statistic allows us in certain cases to perform optimal Bayesian decoding by simple linear transformations, and to obtain closed-form expressions of the Shannon information carried by the network. One technical advantage of the theory is that it may be applied easily even to non-Poisson point process network models; for example, we find that under some conditions, neural populations with strong history-dependent (non-Poisson) effects carry exactly the same information as do simpler equivalent populations of non-interacting Poisson neurons with matched firing rates. We argue that our findings help to clarify some results from the recent literature on neural decoding and neuroprosthetic design.

## Introduction

It has long been argued that many key questions in neuroscience can best be posed in information-theoretic terms; the efficient coding hypothesis discussed in [2, 3, 1], represents perhaps the best-known example. Answering these questions quantitatively requires us to compute the Shannon information rate of neural channels, whether numerically using experimental data or analytically in mathematical models. In many cases it is useful to exploit connections with "ideal observer" analysis, in which the performance of an optimal Bayesian decoder places fundamental bounds on the performance of any biological system given access to the same neural information. However, the non-linear, non-Gaussian, and correlated nature of neural responses has hampered the development of this theory, particularly in the case of high-dimensional and/or time-varying stimuli.

The neural decoding literature is far too large to review systematically here; instead, we will focus our attention on work which has attempted to develop an analytical theory to simplify these complex decoding and information-rate problems. Two limiting regimes have received significant analytical attention in the neuroscience literature. In the "high-SNR" regime, $n \to \infty$, where $n$ is the number of neurons encoding the signal of interest; if the information rate of each neuron is bounded away from zero and neurons respond in a conditionally weakly-dependent manner given the stimulus, then the total information provided by the neural population becomes infinite, and the error rate of any reasonable neural decoder tends to zero. For discrete stimuli, the Shannon information is effectively determined in this asymptotic limit by a simpler quantity known as the Chernoff information [9]; for continuous stimuli, maximum likelihood estimation is asymptotically optimal, and the asymp-

totic Shannon information is controlled by the Fisher information [8, 7]. On the other hand we can consider the "low-SNR" limit, where only a few neurons are observed and each neuron is asymptotically weakly tuned to the stimulus. In this limit, the Shannon information tends to zero, and under certain conditions the optimal Bayesian estimator (which can be strongly nonlinear in general) can be approximated by a simpler linear estimator; see [5] and more recently [16] for details.

In this paper, we study information transmission and optimal decoding in what we would argue is a more biologically-relevant "intermediate" regime, where $n$ is large but the total amount of information provided by the population remains finite, and the problem of decoding the stimulus given the population neural activity remains nontrivial.

## Likelihood in the intermediate regime: the inhomogeneous Poisson case

For clarity, we begin by analyzing the information in a simple population of neurons, represented as inhomogenous Poisson processes that are conditionally independent given the stimulus. We will extend our analysis to more general neural populations in the next section. In response to the stimulus, at each time step $t$ neuron $i$ fires with probability $\lambda_i(t)dt$, where the rate is given by

$$\lambda_i(t) = f\left[b_i(t) + \epsilon\ell_{i,t}(\theta)\right], \tag{1}$$

where $f(.)$ is a smooth rectifying non-linearity and $\epsilon$ is a gain factor controlling each neuron's sensitivity. The baseline firing rate is determined by $b_i(t)$ and is independent of the input signal. The true stimulus at time $t$ is defined by $\theta_t$, and $\theta$ abbreviates the time varying stimulus $\theta_{0:T}$ in the time interval $[0, Tdt]$. The term $\ell_{i,t}(\theta)$ summarizes the dependence of the neuron's firing rate on $\theta$; depending on the setting, this term may represent e.g. a tuning curve or a spatiotemporal filter applied to the stimulus (see examples below).

The likelihood includes all the information about the stimulus encoded in the population's spiking response. Neuron $i$'s response at time step $t$ is designated by by the binary variable $r_i(t)$. The log-likelihood at the parameter value $\vartheta$ (which may be different from the true parameter $\theta$) is given by the standard point-process formula [21]:

$$L_\vartheta(r) := \log p(r|\vartheta) = \sum_{i=1}^{n}\sum_{t=0}^{T} r_i(t)\log\lambda_i(t) - \lambda_i(t)dt. \tag{2}$$

This expression can be expanded around $\epsilon = 0$:

$$L_\vartheta(r) = L_\vartheta(r)|_{\epsilon=0} + \epsilon\frac{\partial L_\vartheta(r)}{\partial\epsilon}|_{\epsilon=0} + \frac{1}{2}\epsilon^2\frac{\partial^2 L_\vartheta(r)}{\partial\epsilon^2}|_{\epsilon=0} + O(n\epsilon^3),$$

where

$$\frac{\partial L_\vartheta(r)}{\partial\epsilon}|_{\epsilon=0} = \sum_{i,t}\ell_{i,t}(\vartheta)\left\{r_i(t)\frac{f'}{f}(b_i(t)) - f'(b_i(t))dt\right\}$$

$$\frac{\partial^2 L_\vartheta(r)}{\partial\epsilon^2}|_{\epsilon=0} = \sum_{i,t}\ell_{i,t}^2(\vartheta)\left\{r_i(t)(\frac{f'}{f})'(b_i(t)) - f''(b_i(t))dt\right\}.$$

Let $r_i$ denote the vector representation of the $i$th neuron's spike train and let[1]

$$g_i(r_i) := \left[r_i(1)\frac{f'}{f}(b_i(1)) - f'(b_i(1))dt \quad \cdots \quad r_i(T)\frac{f'}{f}(b_i(T)) - f'(b_i(T))dt\right]^T$$

$$h_i(r_i) := \left[r_i(1)(\frac{f'}{f})'(b_i(1)) - f''(b_i(1))dt \quad \cdots \quad r_i(T)(\frac{f'}{f})'(b_i(T)) - f''(b_i(T))dt\right]^T$$

$$\ell_i(\vartheta) := \left[\ell_{i,1}(\vartheta)\ \ell_{i,2}(\vartheta)\quad\cdots\quad \ell_{i,T}(\vartheta)\right]^T;$$

then

$$L_\vartheta(r) = L_\vartheta(r)|_{\epsilon=0} + \epsilon\sum_{i=1}^{n}\ell_i(\vartheta)^T g_i(r_i) + \frac{1}{2}\epsilon^2\sum_{i=1}^{n}\ell_i(\vartheta)^T\text{diag}[h_i(r_i)]\ell_i(\vartheta) + O(n\epsilon^3).$$

This second-order loglikelihood expansion is standard in likelihood theory [24]; as usual, the first term is constant in $\vartheta$ and can therefore be ignored, while the third (quadratic) term controls the curvature of the loglikelihood at $\epsilon = 0$, and scales as $\epsilon n^2$. In the high-SNR regime discussed above, where $n \to \infty$ and $\epsilon$ is fixed, the likelihood becomes sharply peaked at $\theta$ (and therefore the Fisher information, which may be understood as the curvature of the log-likelihood at $\theta$, controls the asymptotics of the estimation error in the case of continuous stimuli), and estimation of $\theta$ becomes easy; in the low-SNR regime, we fix $n$ and consider the $\epsilon \to 0$ limit.

Now, finally, we can more precisely define the "intermediate" SNR regime: we will focus on the case of large populations ($n \to \infty$), but in order to keep the total information in a finite range we need to scale the sensitivity $\epsilon$ as $\epsilon \sim n^{-1/2}$. In this setting, the error term $O(n\epsilon^3) = O(n^{-\frac{1}{2}}) = o(1)$ and can therefore be neglected, and the law of large numbers (LLN) implies that

$$\epsilon^2 \frac{\partial^2 L_\vartheta(r)}{\partial \epsilon^2}\Big|_{\epsilon=0} = \mathrm{E}_{r|\theta}\left[\frac{1}{n}\sum_i \ell_i(\vartheta)^T \mathrm{diag}[h_i(r_i)]\ell_i(\vartheta)\right];$$

consequently, the quadratic term $\epsilon^2 \frac{\partial^2 L_\vartheta(r)}{\partial \epsilon^2}\Big|_{\epsilon=0}$ will be independent of the observed spike train and therefore void of information about $\theta$. So the first derivative term is the only part of the likelihood that depends both on the neural activity and $\vartheta$, and may therefore be considered a sufficient statistic in this asymptotic regime: all the information about the stimulus is summarized in

$$\epsilon \frac{\partial L_\vartheta(r)}{\partial \epsilon}\Big|_{\epsilon=0} = \frac{1}{\sqrt{n}}\sum_i \ell_i(\vartheta)^T g_i(r_i). \tag{3}$$

We may further apply the central limit theorem (CLT) to this sum of independent random vectors to conclude that this term converges to a Gaussian process indexed by $\vartheta$ (under mild technical conditions that we will ignore here, for clarity). Thus this model enjoys the local asymptotic normality property observed in many parametric statistical models [24]: all of the information in the data can be summarized asymptotically by a sufficient statistic with a sampling distribution that turns out to be Gaussian.

**Example: Linearly filtered stimuli and state-space models**

In many cases neurons are modeled in terms of simple rectified linear filters responding to the stimulus. We can handle this case easily using the language introduced above, if we let $K_i$ denote the matrix implementing the transformation $(K_i\theta)_t = \ell_{i,t}(\theta)$, the projection of the stimulus onto the $i$-th neuron's stimulus filter. Then,

$$\epsilon \frac{\partial L_\vartheta(r)}{\partial \epsilon}\Big|_{\epsilon=0} = \vartheta^T\left[\frac{1}{\sqrt{n}}\sum_{i=1}^n K_i^T\left(\mathrm{diag}\left[\frac{f_i'}{f_i}\right]r_i - f_i'dt\right)\right] := \vartheta^T\Delta(r),$$

where $f_i$ stands for the vector version of $f[b_i(t)]$. Thus all the information in the population spike train can be summarized in the random vector $\Delta(r)$, which is a simple linear function of the observed spike train data. This vector has an asymptotic Gaussian distribution, with mean and covariance

$$\mathrm{E}_{r|\theta}\left(\Delta(r)\right) = \frac{1}{\sqrt{n}}\sum_{i=1}^n K_i^T\left(\mathrm{diag}\left[\frac{f_i'}{f_i}\right]\left(f_i dt + f_i'dt\frac{K_i\theta}{\sqrt{n}} + O(\frac{1}{n})\right) - f_i'dt\right)$$

$$= \left[\frac{1}{n}\sum_{i=1}^n K_i^T\mathrm{diag}\left[\frac{f_i'^2}{f_i}dt\right]K_i\right]\theta + O(\frac{1}{\sqrt{n}})$$

$$J := \mathrm{cov}_{r|\theta}\left(\Delta(r)\right) = \frac{1}{n}\sum_{i=1}^n K_i^T\mathrm{diag}\left[\frac{f_i'}{f_i}\right]\mathrm{cov}_{r|\theta}\left[r_i\right]\mathrm{diag}\left[\frac{f_i'}{f_i}\right]K_i$$

$$= \frac{1}{n}\sum_{i=1}^n K_i^T\mathrm{diag}\left[\frac{f_i'^2}{f_i}dt\right]K_i + O(\frac{1}{\sqrt{n}}).$$

Thus, the neural population's non-linear and temporally dynamic response to the stimulus is as informative in this intermediate regime as a single observation from a standard Gaussian experiment,

in which the parameter $\theta$ is filtered linearly by $J$ and corrupted by Gaussian noise. All of the filtering properties of the population are summarized by the matrix $J$. (Note that if we consider each $K_i$ as a random sample from some distribution of filters, then $J$ will converge by the law of large numbers to a matrix we can compute explicitly.)

Thus in many cases we can perform optimal Bayesian decoding of $\theta$ given the spike trains quite easily. For example, if $\theta$ has a zero mean Gaussian prior distribution with covariance $C_\theta$, then the posterior mean and the maximum-a-posteriori (MAP) estimate is well-known and coincides with the optimal linear estimate (OLE):

$$\hat{\theta}_{OLE}(r) = E(\theta|r) = (J + C_\theta^{-1})^{-1}\Delta(r). \tag{4}$$

We may compute the Shannon information $I(\theta : r)$ between $r$ and $\theta$ in a similarly direct fashion. We know that, asymptotically, the sufficient statistic $\Delta(r)$ is as informative as the full population response $r$

$$I(\theta : r) = I(\theta : \Delta(r)).$$

In the case that the prior of $\theta$ is Gaussian, as above, then the information can therefore be computed quite explicitly via standard formulas for the linear-Gaussian channel [9]:

$$I(\theta : \Delta(r)) = \frac{1}{2}\log\det(I + JC_\theta). \tag{5}$$

To summarize, when the encodings $\ell_{i,t}(\theta)$ are linear in $\theta$, and we are in the intermediate-SNR regime, and the parameter $\theta$ has a Gaussian prior distribution, then the optimal Bayesian estimate is obtained by applying a linear transformation to the sufficient statistic $\Delta(r)$ which itself is linear in the spike train, and the mutual information between the stimulus and full population response has a particularly simple form. These results help to extend previous theoretical studies [5, 18, 20, 16] demonstrating that in some cases linear decoding can be optimal, and also shed some light on recent experimental studies indicating that optimal linear and nonlinear Bayesian estimators often have similar performance in practice [13, 12].

To work through a concrete example, consider the case that the temporal sequence of parameter values $\theta_t$ is generated by an autoregressive process:

$$\theta_{t+1} \quad = \quad A\theta_t + \eta_t \quad \eta_t \sim \mathcal{N}(0, R),$$

for a stable dynamics matrix $A$ and positive-semidefinite covariance matrix $R$. Further assume that the observation matrices $K_i$ act instantaneously, i.e., $K_i$ is block-diagonal with blocks $K_{i,t}$, and therefore the responses are modeled as

$$r_i(t) \sim Poiss[f(b_i(t) + \epsilon K_{i,t}\theta_t)dt].$$

Thus $\theta$ and the responses $r$ together represent a state-space model. This framework has been shown to lead to state-of-the-art performance in a wide variety of neural data analysis settings [14]. To understand optimal inference in this class of models in the intermediate SNR regime, we may follow the recipe outlined above: we see that the asymptotic sufficient statistic in this model can be represented as

$$\Delta_t = J_t\theta_t + \epsilon_t \quad \epsilon_t \sim \mathcal{N}(0, J_t),$$

where the effective filter matrix $J$ defined above is block-diagonal (due to the block-diagonal structure of the filter matrices $K_i$), with blocks we have denoted $J_t$. Thus $\Delta_t$ represents observations from a linear-Gaussian state-space model, i.e., a Kalman filter model [17]. Optimal decoding of $\theta$ given the observation sequence $\Delta_{1:T}$ can therefore be accomplished via the standard forward-backward Kalman filter-smoother [10]; see Fig. 1 for an illustration. The information rate $\lim_{T\to\infty} I(\theta_{0:T} : r_{0:T}) = \lim_{T\to\infty} I(\theta_{0:T} : \Delta(r)_{0:T})$ may be computed via similar recursions in the stationary case (i.e., when $J_t$ is constant in time). The result may be expressed most explicitly in terms of a matrix which is the solution of a Riccati equation involving the effective Kalman model parameters; the details are provided in the appendix.

**Nonlinear examples: orientation coding, place fields, and small-time expansions**

While the linear setting discussed above can handle many examples of interest, it does not seem general enough to cover two well-studied decoding problems: inferring the orientation of a visual

stimulus from a population of cortical neurons [19, 4], or inferring position from a population of hippocampal or entorhinal neurons [6]. In the former case, the stimulus is a phase variable, and therefore does not fit gracefully into the linear setting described above; in the latter case, place fields and grid fields are not well-approximated as linear functions of position. If we apply our general theory in these settings, the interpretation of the encoding function $\ell_i(\theta)$ does not change significantly: $\ell_i(\theta)$ could represent the tuning curve of neuron $i$ as a function of the orientation of the visual stimulus, or of the animal's location in space. However, without further assumptions the limiting sufficient statistic, which is a weighted sum of these encoding functions $\ell_i(\theta)$ (recall eq. 3) may result in an infinite-dimensional Gaussian process, which may be computationally inconvenient.

To simplify matters somewhat, we can introduce a mild assumption on the tuning functions $\ell_i(\theta)$. Let's assume that these functions may be expressed in some low-dimensional basis: $\ell_i(\theta) = K_i\Phi(\theta)$, for some vectors $K_i$, and $\Phi(\theta)$ is defined to map $\theta$ into an $mT$-dimensional space which is usually smaller than $\dim(\theta) = \dim(\theta_t)T$. This finite-basis assumption is very natural: in the orientation example, tuning curves are periodic in the angle $\theta_t$ and are therefore typically expressed as sums of a few Fourier functions; similarly, two-dimensional finite Fourier or Zernike bases are often used to represent grid or place fields [6]. The key point here is that we may now simply follow the derivation of the last section with $\Phi(\theta)$ in place of $\theta$; we find that the sufficient statistic may be represented asymptotically as an $mT$-dimensional Gaussian vector with mean $J$ and covariance $J\Phi(\theta)$, with $J$ defined as in the preceding section.

We should note that this nonlinear case does remain slightly more complicated than the linear case in one respect: while the likelihood with respect to $\Phi(\theta)$ reduces to something very simple and tractable, the prior (which is typically defined as a function of $\theta$) might be some complicated function of the remapped variable $\Phi(\theta)$. So in most interesting nonlinear cases we can no longer compute the optimal Bayesian decoder or the Shannon information rate analytically. However, our approach does lead to a major simplification in numerical investigations into theoretical coding issues. For example, to examine the coding efficiency of a population of neurons encoding an orientation variable in this intermediate SNR regime we do not need to simulate the responses of the entire population (which would involve drawing $nT$ random variables, for some large population size $n$); instead, we only need to draw a single equivalent $mT$-dimensional Gaussian vector $\Delta(r)$, and quantify the decoding performance based on the approximate loglikelihood

$$L_\vartheta(r) = L_\vartheta(r)|_{\epsilon=0} + \Phi(\vartheta)^T\Delta(r) + \frac{1}{2}\Phi(\vartheta)^T J\Phi(\vartheta) + O(\frac{1}{\sqrt{n}}),$$

which as emphasized above has a simple quadratic form as a function of $\Phi(\vartheta)$. Since $m$ can typically be chosen to be much smaller than $n$, this approach can result in significant computational savings.

We now switch gears slightly and examine another related intermediate regime in which nonlinear encoding plays a key role: instead of letting the sensitivity $\epsilon$ of each neuron become small (in order to keep the total information in the population finite), we could instead keep the sensitivity constant and let the time period over which we are observing the population scale inversely with the population size $n$. This short-time limit is sensible in some physiological and psychophysical contexts [22] and was examined analytically in [15] to study the impact of inter-neuron dependencies on information transmission. Our methods can also be applied to this short-time limit. We begin by writing the loglikelihood of the observed spike count vector $r$ in a single time-bin of length $dt$:

$$L_\vartheta(r) := \log p(r|\theta) = \sum_i r_i \log f[b_i + \ell_i(\vartheta)] - f[b_i + \ell_i(\vartheta)]\,dt.$$

The second term does not depend on $r$; therefore, all information in $r$ about $\theta$ resides in the sufficient statistic

$$\Delta_\vartheta(r) := \sum_i r_i \log f[b_i + \ell_i(\vartheta)].$$

Since the $i$-th neuron fires with probability $f[b_i + \ell_i(\theta)]\,dt$, the mean of $\Delta_\vartheta(r)$ scales with $ndt$, and it is clear that $dt = 1/n$ is a natural scaling of the time bin. With this scaling $\Delta_\vartheta(r)$ converges to a Gaussian stochastic process with mean

$$E_{r|\theta}[\Delta_\vartheta(r)] = \frac{1}{n}\sum_i f[b_i + \ell_i(\theta)]\log f[b_i + \ell_i(\vartheta)]$$

and covariance

$$\text{cov}_{r|\theta}[\Delta_\vartheta(r), \Delta_{\vartheta'}(r)] = \frac{1}{n}\sum_i f\left[b_i + \ell_i(\theta)\right]\left(\log f\left[b_i + \ell_i(\vartheta)\right]\right)\left(\log f\left[b_i + \ell_i(\vartheta')\right]\right),$$

where we have used the fact that the variance of a Poisson random variable coincides with its mean.

In general, this limiting Gaussian process will be infinite-dimensional. However, if we choose the exponential nonlinearity ($f(.) = \exp(.)$) and the encoding functions $\ell_i(\theta)$ are of the finite-dimensional form considered above, $\ell_i(\theta) = K_i^T\Phi(\theta)$, then the $\log f[b_i + \ell_i(\vartheta)]$ term in the definition of $\Delta_\vartheta(r)$ simplifies: in this case, all information about $\theta$ is captured by the sufficient statistic

$$\Delta(r) = \sum_i r_i K_i.$$

If we again let $dt = 1/n$, then we find that $\Delta(r)$ converges to a finite-dimensional Gaussian random vector with mean and covariance

$$E_{r|\theta}[\Delta(r)] = \frac{1}{n}\sum_i f\left[b_i + K_i^T\Phi(\theta)\right]K_i; \qquad \text{cov}_{r|\theta}[\Delta(r)] = \frac{1}{n}\sum_i f\left[b_i + K_i^T\Phi(\theta)\right]K_i K_i^T;$$

again, if the filters $K_i$ are modeled as independent draws from some fixed distribution, then the above normalized sums converge to their expectations, by the LLN. Thus, as in the intermediate-SNR regime, we see that inference can be dramatically simplified in this short-time setting.

## Likelihood in the intermediate regime: non-Poisson effects

We conclude by discussing the generalization to non-Poisson networks with interneuronal dependencies and nontrivial correlation structure. We generalize the rate equation (1) to

$$\lambda_i(t) = f_i\left[b_i(t) + \epsilon\ell_{i,t}(\theta)\big|\mathcal{H}_t\right],$$

where $\mathcal{H}_t$ stands for the spiking activity of all neurons prior to time $t$: $\mathcal{H}_t = \{r_i(t')\}_{t'<t, 1\le i\le n}$. Note that the influence of spiking history may be different for each neuron: refractory periods, self-inhibition and coupling between neurons can be formulated by appropriately defining the dependence of $f_i(.)$ on $\mathcal{H}_t$.

We begin, as usual, by expanding the log-likelihood. The basic point-process likelihood (eq. 2) remains valid. Let $g_i(r)$ and $h_i(r)$ denote the vector versions of

$$r_i(t)\frac{f'}{f}\left[b_i(t)\big|\mathcal{H}_t\right] - f_i'\left[b_i(t)\big|\mathcal{H}_t\right]dt \quad \text{and} \quad r_i(t)\left(\frac{f'}{f}\right)'\left[b_i(t)\big|\mathcal{H}_t\right] - f_i''\left[b_i(t)\big|\mathcal{H}_t\right]dt,$$

respectively, analogously to the Poisson case. Then, the first and second terms in the expansion of the loglikelihood may be written as

$$\epsilon\frac{\partial L_\vartheta(r)}{\partial\epsilon}\big|_{\epsilon=0} = \epsilon\sum_i \ell_i^T(\vartheta)g_i(r) \quad \text{and} \quad \frac{1}{2}\epsilon^2\frac{\partial^2 L_\vartheta(r)}{\partial\epsilon^2}\big|_{\epsilon=0} = \frac{1}{2}\epsilon^2\sum_i \ell_i^T(\vartheta)\text{diag}[h_i(r)]\ell_i(\vartheta),$$

as before. For independent neurons, the log-likelihood was composed of normalized sums of independent random variables that converged to a Gaussian process, by the CLT. In the history-dependent, coupled case, $g_i(r)$ and $h_i(r)$ depend not only on the $i$-th neuron's activity $r_i$, but rather on the whole network history. Nonetheless, under technical conditions on the network's dependence structure (to ensure that the firing rates and correlations in the network remain bounded), we may still exploit versions of the LLN and CLT. Thus, under conditions ensuring the validity of the LLN we may conclude that, as before, the second-order term $\epsilon^2\frac{\partial^2 L_\vartheta(r)}{\partial\epsilon^2}\big|_{\epsilon=0}$ converges to its expectation under the intermediate $\epsilon \sim n^{-\frac{1}{2}}$ scaling, and therefore carries no information about $\theta$. When we discard this second-order term, along with higher-order terms that are negligible in the intermediate-SNR, large-$n$ limit, we are left once again with the gradient term $\epsilon\frac{\partial L_\vartheta(r)}{\partial\epsilon}\big|_{\epsilon=0} = \frac{1}{\sqrt{n}}\sum_i \ell_i(\vartheta)^T g_i(r)$, which under appropriate conditions (ensuring the validity of a CLT) will converge to a Gaussian process limit whose mean and covariance we can often compute analytically.

Let's turn to a specific example, in order to make these claims somewhat more concrete. Consider a network with weak couplings and possibly strong self-inhibition and history dependence; more precisely, we assume that interneuronal conditional cross-covariances are weak, given the stimulus:

$$\text{cov}[r_i(t), r_j(t+\tau)|\theta] = O(n^{-1}) \quad \text{for } i \neq j.$$

See, e.g., [11, 23] for further discussion of this condition, which is satisfied for many spiking networks in which the synaptic weights scale uniformly as $O(n^{-1})$. For simplicity, we will also restrict our attention to linear encoding functions, though generalizations to the nonlinear case are straightforward. Thus, as before, let $K_i$ denote the matrix implementing the transformation $(K_i\theta)_t = \ell_{i,t}(\theta)$, the projection of the stimulus onto the $i$-th neuron's stimulus filter. Then

$$\epsilon \frac{\partial L_\vartheta(r)}{\partial \epsilon}\Big|_{\epsilon=0} = \vartheta^T \left[ \frac{1}{\sqrt{n}} \sum_{i=1}^n K_i^T \left( \text{diag}\left[\frac{f_i'}{f_i}\right] r_i - f_i'dt \right) \right],$$

where $f_i$ stands for the vector version of $f_i\left[b_i(t)\big|\mathcal{H}_t\right]$; in other words, the $t$-th entry of $f_idt$ is the probability of observing a spike in the interval $[t, t+dt]$, given the network spiking history $\mathcal{H}_t$ in the absence of input. Our sufficient statistic is therefore exactly as in the Poisson setting,

$$\Delta(r) := \frac{1}{\sqrt{n}} \sum_{i=1}^n K_i^T \left( \text{diag}\left[\frac{f_i'}{f_i}\right] r_i - f_i'dt \right), \tag{6}$$

except for the history-dependence induced through the redefinition of $f_i$.

Computing the necessary means and covariances in this case requires more work than in the Poisson case; see the appendix for details. It is helpful (though not necessary) to make the stationarity assumption $b_i(t) \equiv b_i$, which implies in this setting that $E(\frac{f_i'^2}{f_i})$ can also be chosen to be time-invariant; in this case the limiting covariance and mean of the sufficient statistic are given by

$$J := \text{cov}_{r|\theta}[\Delta(r)] = \frac{1}{n} \sum_{i=1}^n K_i \text{diag}\left[ E_{r|\theta=0}(\frac{f_i'^2}{f_i}dt) \right] K_i; \quad E_{r|\theta}[\Delta(r)] = J\theta,$$

where the expectations are over the spontaneous network activity in the absence of any input. In short, once again, we have $\Delta(r) \to_D \mathcal{N}(J\theta, J)$. Analytically, the only challenge here is to compute the expectations in the definition of $J$. In many cases this can be done analytically (e.g., in any population of uncoupled renewal-process neurons), or by using mean-field theory [23], or numerically by simply calculating the mean firing rate of the network in the undriven state $\theta = 0$.

We examine this convergence quantitatively in Fig. 1. In this case the stimulus $\theta_t$ was a sample path from a one-dimensional autoregressive (AR(1)) process. Spikes were generated according to

$$\lambda_i(t) = \lambda_o \exp\left( \frac{\theta_t}{\sqrt{n}} + \sum_{j=1}^n w_{ji}I_j(t) \right) 1_{\tau_i(t)>\tau_{\text{ref}}},$$

where $I_j(t)$ is the synaptic input from the $j$-th cell (generated by convolving the spike train $r_j$ with an exponential of time constant 20 ms), $w_{ji}$ is the synaptic weight matrix coupling the output of neuron $j$ to the input of neuron $i$, $\tau_i(t)$ is the time since the last spike; therefore, $1_{\tau_i(t)>\tau_{\text{ref}}}$ enforces the absolute refractory period $\tau_{\text{ref}}$, which was set to be 2 ms here. Since the encoding filters $K_i$ act instantaneously in this model ($K_i$ can be represented as a delta function, weighted by $n^{-1/2}$), the observed spike trains can be considered observations from a state-space model, as described above. The weights $w_{ji}$ were generated randomly from a uniform distribution on the interval $-[5/n, 5/n]$, with self-weights $w_{ii} = 0$, and $\sum_j w_{ji} = 0$ to enforce detailed balance in the network. Note that, while the interneuronal coupling is weak in this example, the autocorrelation in these spike trains is quite strong on short time scales, due to the absolute refractory effect.

We compared two estimators of $\theta$: the full (nonlinear) MAP estimate $\hat{\theta}_{MAP} = \arg\max_\theta p(\theta|r)$, which we computed using the fast direct optimization methods described in [14], and the limiting optimal estimator $\hat{\theta}_\Delta := (J + C_\theta^{-1})^{-1}\Delta(r)$. Note that $J$ is diagonal; we computed the expectations in the definition of $J$ using the numerical approach described above in this simulation, though in

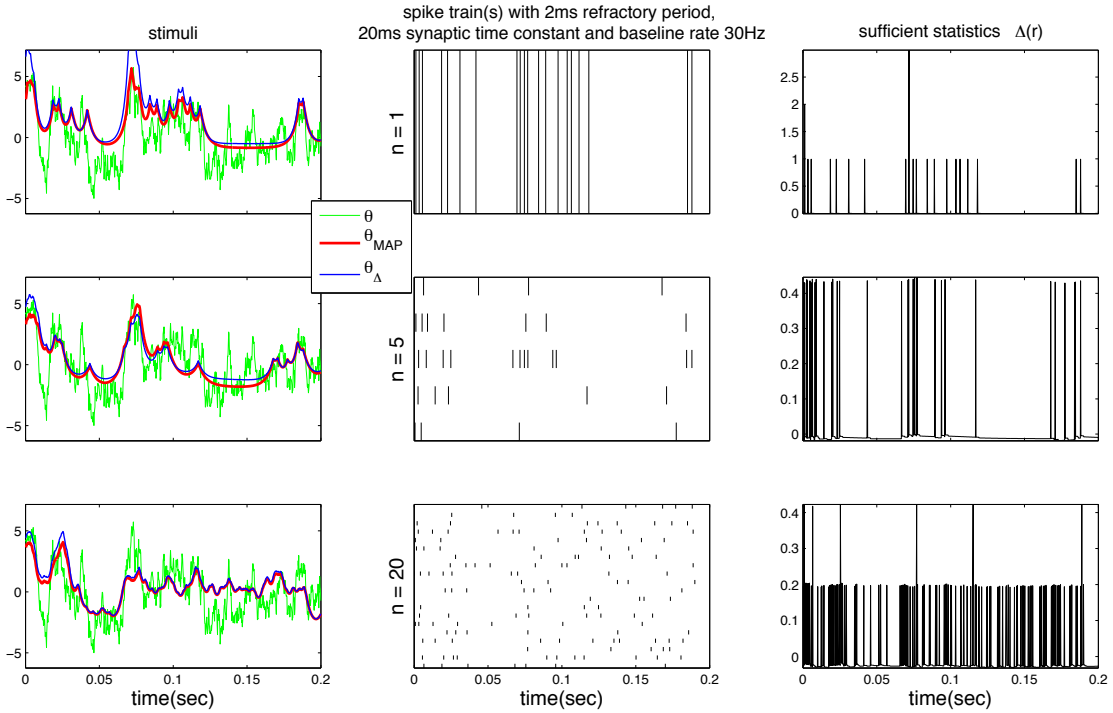

Figure 1: The left panels show the true stimulus (green), MAP estimate (red) and the limiting optimal estimator $\hat{\theta}_{\Delta} := (J + C_{\theta}^{-1})^{-1}\Delta(r)$ (blue) for various population sizes $n$. The middle panels show the spike trains used to compute these estimates. The right panels show the sufficient statistics $\Delta(r)$ used to compute $\hat{\theta}_{\Delta}$. Note that the same true stimulus was used in all three simulations. As $n$ increases, the linear decoder converges to the MAP estimate, despite the nonlinear and correlated nature of the network model generating the spike trains (see main text for details).

other simulations (with uncoupled renewal-model populations) we checked that the fully-analytical approach gave the correct solution. In addition, $C_{\theta}^{-1}$ is tridiagonal in this state-space setting; thus the linear matrix equation in eq. (4) can be solved efficiently in $O(T)$ time using standard tridiagonal matrix solvers. We find that, as predicted, the full nonlinear Bayesian estimator $\hat{\theta}_{MAP}$ approaches the limiting optimal estimator $\hat{\theta}_{\Delta}$ as $n$ becomes large; $n = 20$ is basically sufficient in this case, although of course the convergence will be slower for larger values of the gain factor $\epsilon$ (or, equivalently, larger filters $K_i$ or larger values of the variance of $\theta_t$).

We conclude with a few comments about these results. First, note that the covariance matrix $J$ we have computed here coincides almost exactly with what we computed previously in the Poisson case. Indeed, we can make this connection much more precise: we can always choose an equivalent Poisson network with rates defined so that the $\mathrm{E}_{r|\theta=0}[(f_i')^2/f_i]$ term in the non-Poisson network matches the $(f_i')^2/f_i$ term in the Poisson network. Since $J$ determines the information rate completely, we conclude that for any weakly-coupled network there is an equivalent Poisson network which conveys exactly the same information in the intermediate regime. However, note that the the sufficient statistic $\Delta(r)$ is different in the Poisson and non-Poisson settings, since the $f'/f$ term linearly reweights the observed spikes, depending on how likely they were given the history; thus the optimal Bayesian decoder incorporates non-Poisson effects explicitly.

A number of interesting questions remain open. For example, while we expect a LLN and CLT to continue to hold in many cases of strong, structured interneuronal coupling, computing the asymptotic mean and covariance of the sufficient statistic $\Delta(r)$ may be more challenging in such cases, and new phenomena may arise.

## Footnotes

[1]With a slight abuse of notation, we use $T$ for both the total number of time steps and the transpose operation; the difference is clear from the context.

# References

[1] J. Atick. Could information theory provide an ecological theory of sensory processing? *Network: Computation in Neural Systems*, pages 213–251, May 1992.

[2] F. Attneave. Some informational aspects of visual perception. *Psychological Review*, 1954.

[3] H. B. Barlow. Possible principles underlying the transformation of sensory messages. *Sensory Communication*, pages 217–234, 1961.

[4] P. Berens, A. S. Ecker, S. Gerwinn, A. S. Tolias, and M. Bethge. Reassessing optimal neural population codes with neurometric functions. *Proceedings of the National Academy of Sciences*, 108:4423–4428, 2011.

[5] W. Bialek and A. Zee. Coding and computation with neural spike trains. *Journal of Statistical Physics*, 59:103–115, 1990.

[6] E. Brown, L. Frank, D. Tang, M. Quirk, and M. Wilson. A statistical paradigm for neural spike train decoding applied to position prediction from ensemble firing patterns of rat hippocampal place cells. *Journal of Neuroscience*, 18:7411–7425, 1998.

[7] N. Brunel and J.-P. Nadal. Mutual information, fisher information, and population coding. *Neural Comput.*, 10(7):1731–1757, 1998.

[8] B. Clarke and A. Barron. Information-theoretic asymptotics of Bayes methods. *IEEE Transactions on Information Theory*, 36:453 – 471, 1990.

[9] T. Cover and J. Thomas. *Elements of information theory*. Wiley, New York, 1991.

[10] J. Durbin and S. Koopman. *Time Series Analysis by State Space Methods*. Oxford University Press, 2001.

[11] I. Ginzburg and H. Sompolinsky. Theory of correlations in stochastic neural networks. *Phys Rev E*, 50(4):3171–3191, 1994.

[12] V. Lawhern, W. Wu, N. Hastopoulos, and L. Paninski. Population decoding of motor cortical activity using a generalized linear model with hidden states. *Journal of Neuroscience Methods*, 2011.

[13] J. Macke, L. Sing, B. Cunningham, J.P. snd Yu, K. Shenoy, and M. Sahani. Modelling low-dimensional dynamics in recorded spiking populations. *COSYNE*, 2011.

[14] L. Paninski, Y. Ahmadian, D. Ferreira, S. Koyama, K. Rahnama Rad, M. Vidne, J. Vogelstein, and W. Wu. A new look at state-space models for neural data. *Journal of Computational Neuroscience*, 29(1):107–126, 2010.

[15] S. Panzeri, S. Schultz, A. Treves, and E. Rolls. Correlations and the encoding of information in the nervous system. *Proceedings of the Royal Society London B*, 266(1423):1001–1012, 1999.

[16] J. Pillow, Y. Ahmadian, and L. Paninski. Model-based decoding, information estimation, and change-point detection in multi-neuron spike trains. *Neural Computation*, 23(1):1–45, January 2011.

[17] S. Roweis and Z. Ghahramani. A unifying review of linear Gaussian models. *Neural Computation*, 11:305–345, 1999.

[18] E. Salinas and L. Abbott. Vector reconstruction from firing rates. *Journal of Computational Neuroscience*, 1:89–107, 1994.

[19] H. S. Seung and H. Sompolinsky. Simple models for reading neuronal population codes. *Proceedings of the National Academy of Sciences*, 90:10749–10753, 1993.

[20] H. Snippe. Parameter extraction from population codes: A critical assesment. *Neural Computation*, 8:511–529, 1996.

[21] D. Snyder and M. Miller. *Random Point Processes in Time and Space*. Springer-Verlag, 1991.

[22] S. Thorpe, D. Fize, and C. Marlot. Speed of processing in the human visual system. *Nature*, 381:520–522, 1996.

[23] T. Toyoizumi, K. Rahnama Rad, and L. Paninski. Mean-field approximations for coupled populations of generalized linear model spiking neurons with Markov refractoriness. *Neural Computation*, 21:1203–1243, 2009.

[24] A. van der Vaart. *Asymptotic statistics*. Cambridge University Press, Cambridge, 1998.

